# REFLEXIVE ASSOCIATIVE MEMORIES

Hendricus G. Loos
Laguna Research Laboratory, Fallbrook, CA 92028-9765

## ABSTRACT

In the synchronous discrete model, the average memory capacity of bidirectional associative memories (BAMs) is compared with that of Hopfield memories, by means of a calculation of the percentage of good recall for 100 random BAMs of dimension 64x64, for different numbers of stored vectors. The memory capacity is found to be much smaller than the Kosko upper bound, which is the lesser of the two dimensions of the BAM. On the average, a 64x64 BAM has about 68 % of the capacity of the corresponding Hopfield memory with the same number of neurons. Ortho-normal coding of the BAM increases the effective storage capacity by only 25 %. The memory capacity limitations are due to spurious stable states, which arise in BAMs in much the same way as in Hopfield memories. Occurrence of spurious stable states can be avoided by replacing the thresholding in the backlayer of the BAM by another nonlinear process, here called "Dominant Label Selection" (DLS). The simplest DLS is the winner-take-all net, which gives a fault-sensitive memory. Fault tolerance can be improved by the use of an orthogonal or unitary transformation. An optical application of the latter is a Fourier transform, which is implemented simply by a lens.

## INTRODUCTION

A reflexive associative memory, also called bidirectional associative memory, is a two-layer neural net with bidirectional connections between the layers. This architecture is implied by Dana Anderson's optical resonator[1], and by similar configurations[2,3]. Bart Kosko[4] coined the name "Bidirectional Associative Memory" (BAM), and investigated several basic properties[4-6]. We are here concerned with the memory capacity of the BAM, with the relation between BAMs and Hopfield memories[7], and with certain variations on the BAM.

## BAM STRUCTURE

We will use the discrete model in which the state of a layer of neurons is described by a bipolar vector. The Dirac notation[8] will be used, in which |> and <| denote respectively column and row vectors. <a| and |a> are each other transposes, <a|b> is a scalar product, and |a><b| is an outer product. As depicted in Fig. 1, the BAM has two layers of neurons, a front layer of N neurons with state vector |f>, and a back layer

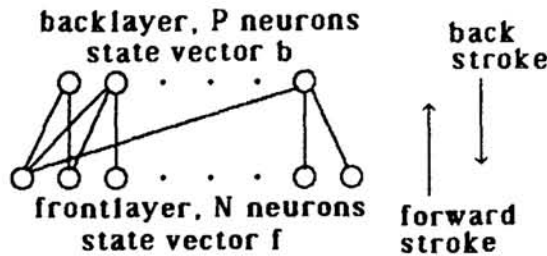

Fig. 1. BAM structure

of P neurons with state vector |b>. The bidirectional connections between the layers allow signal flow in two directions. The front stroke gives |b>= s(B|f>), where B is the connection matrix, and s( ) is a threshold function, operating at zero. The back stroke results in an upgraded front state <f'|=s(<b|B), which also may be written as |f'>=s(B$^T$|b>), where the superscript T denotes transposition. We consider the synchronous model, where all neurons of a layer are updated simultaneously, but the front and back layers are updated at different times. The BAM action is shown in Fig. 2. The forward stroke entails taking scalar products between a front state vector |f> and the rows of B, and entering the thresholded results as elements of the back state vector |b>. In the back stroke we take

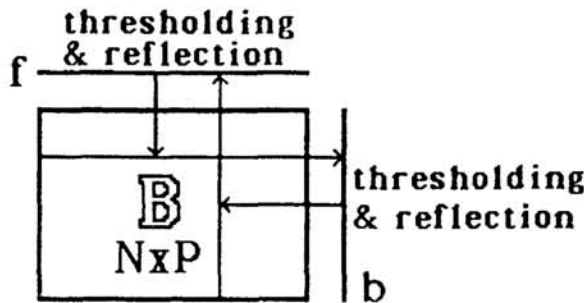

Fig. 2. BAM action

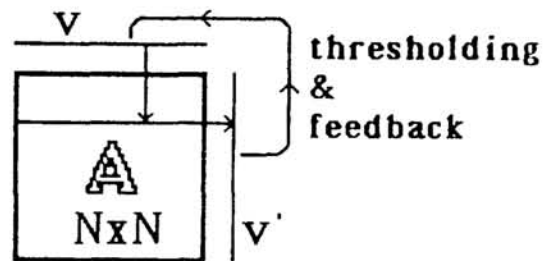

Fig. 3. Autoassociative memory action

scalar products of |b> with column vectors of B, and enter the thresholded results as elements of an upgraded state vector |f'>. In contrast, the action of an autoassociative memory is shown in Figure 3. The BAM may also be described as an autoassociative memory[5] by

concatenating the front and back vectors into a single state vector $|v\rangle = |f,b\rangle$, and by taking the $(N+P) \times (N+P)$ connection matrix as shown in Fig. 4. This autoassociative memory has the same number of neurons as our

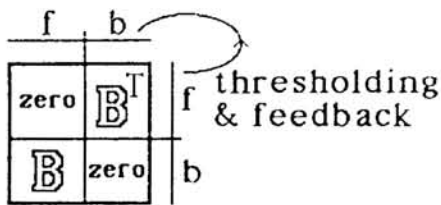

Fig. 4. BAM as autoasso-
ciative memory

BAM, viz. $N+P$. The BAM operation where initially only the front state is speci- fied may be obtained with the corres- ponding autoassociative memory by initially specifying $|b\rangle$ as zero, and by arranging the thresholding operation such that $s(0)$ does not alter the state vector component. For a Hopfield memory[7] the connection matrix is

$$H = \left( \sum_{m=1}^{M} |m\rangle\langle m| \right) - MI \ , \tag{1}$$

where $|m\rangle$, $m=1$ to $M$, are stored vectors, and $I$ is the identity matrix. Writing the $N+P$ dimensional vectors $|m\rangle$ as concatenations $|d_m, c_m\rangle$, (1) takes the form

$$H = \left( \sum_{m=1}^{M} (|d_m\rangle\langle d_m| + |c_m\rangle\langle c_m| + |d_m\rangle\langle c_m| + |c_m\rangle\langle d_m|) \right) - MI \ , \tag{2}$$

with proper block placing of submatrices understood. Writing

$$K = \sum_{m=1}^{M} |c_m\rangle\langle d_m| \ , \tag{3}$$

$$H_d = \left( \sum_{m=1}^{M} |d_m\rangle\langle d_m| \right) - MI \ , \qquad H_c = \left( \sum_{m=1}^{M} |c_m\rangle\langle c_m| \right) - MI, \tag{4}$$

where the $I$ are identities in appropriate subspaces, the Hopfield matrix $H$ may be partitioned as shown in Fig. 5. $K$ is just the BAM matrix given by Kosko[5], and previously used by Kohonen[9] for linear heteroassociative memories. Comparison of Figs. 4 and 5 shows that in the synchronous discrete model the BAM with connection matrix (3) is equivalent to a Hopfield memory in which the diagonal blocks $H_d$ and $H_c$ have been

deleted. Since the Hopfield memory is robust, this "pruning" may not affect much the associative recall of stored vectors, if M is small; however, on the average, pruning will not improve the memory capacity. It follows that, on the average, a discrete synchronous BAM with matrix (3) can at best have the capacity of a Hopfield memory with the same number of neurons.

We have performed computations of the average memory capacity for 64x64 BAMs and for corresponding 128x128 Hopfield memories. Monte Carlo calculations were done for 100 memories, each of which stores M random bipolar vectors. The straight recall of all these vectors was checked, allowing for 24 iterations. For the BAMs, the iterations were started with a forward stroke in which one of the stored vectors $|d_m\rangle$ was used as input. The percentage of good recall and its standard deviation were calculated. The results plotted in Fig. 6 show that the square BAM has about 68% of the capacity of the corresponding Hopfield memory. Although the total number of neurons is the same, the BAM only needs 1/4 of the number of connections of the Hopfield memory. The storage capacity found is much smaller than the Kosko [6] upper bound, which is min (N,P).

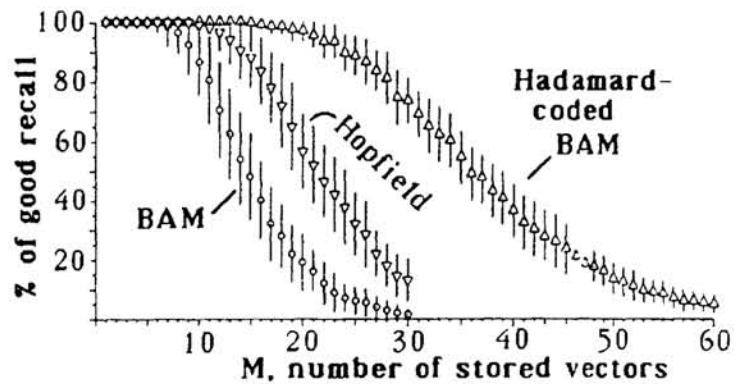

$$\mathbb{H} = \begin{array}{|c|c|} \hline \mathbb{H}_d & \mathbb{K}^T \\ \hline \mathbb{K} & \mathbb{H}_c \\ \hline \end{array}$$

Fig. 5. Partitioned Hopfield matrix

Fig. 6. % of good recall versus M

CODED BAM

So far, we have considered both front and back states to be used for data. There is another use of the BAM in which only front states are used as data, and the back states are seen as providing a code, label, or pointer for the front state . Such use was anticipated in our expression (3) for the BAM matrix which stores data vectors $|d_m\rangle$ and their labels or codes $|c_m\rangle$. For a square BAM, such an arrangement cuts the information contained in a single stored data vector in half. However, the freedom of

choosing the labels $|c_m\rangle$ may perhaps be put to good use. Part of the problem of spurious stable states, which plagues BAMs as well as Hopfield memories as they are loaded up, is due to the lack of orthogonality of the stored vectors. In the coded BAM we have the opportunity to remove part of this problem by choosing the labels as orthonormal. Such labels have been used previously by Kohonen[9] in linear heteroassociative memories. The question whether memory capacity can be improved in this manner was explored by taking 64x64 BAMs in which the labels are chosen as Hadamard vectors. The latter are bipolar vectors with Euclidean norm $\sqrt{P}$, which form an orthonormal set. These vectors are rows of a PxP Hadamard matrix; for a discussion see Harwit and Sloane[10]. The storage capacity of such Hadamard-coded BAMs was calculated as function of the number M of stored vectors for 100 cases for each value of M, in the manner discussed before. The percentage of good recall and its standard deviation are shown in Fig. 6. It is seen that the Hadamard coding gives about a factor 2.5 in M, compared to the ordinary 64x64 BAM. However, the coded BAM has only half the stored data vector dimension. Accounting for this factor 2 reduction of data vector dimension, the effective storage capacity advantage obtained by Hadamard coding comes to only 25 %.

## HALF BAM WITH HADAMARD CODING

For the coded BAM there is the option of deleting the threshold operation in the front layer. The resulting architecture may be called "half BAM". In the half BAM, thresholding is only done on the labels, and consequently, the data may be taken as analog vectors. Although such an arrangement diminishes the robustness of the memory somewhat, there are applications of interest. We have calculated the percentage of good recall for 100 cases, and found that giving up the data thresholding cuts the storage capacity of the Hadamard-coded BAM by about 60 %.

## SELECTIVE REFLEXIVE MEMORY

The memory capacity limitations shown in Fig. 6 are due to the occurence of spurious states when the memories are loaded up.

Consider a discrete BAM with stored data vectors $|m\rangle$, m=1 to M, orthonormal labels $|c_m\rangle$, and the connection matrix

$$K = \sum_{m=1}^{M} |c_m\rangle\langle m| \quad . \tag{5}$$

For an input data vector $|v\rangle$ which is closest to the stored data vector $|1\rangle$, one has in the forward stroke

$$|b\rangle = s(c|c_1\rangle + \sum_{m=2}^{M} a_m|c_m\rangle) \quad , \tag{6}$$

where

$$c = \langle 1|v\rangle \quad , \quad \text{and} \quad a_m = \langle m|v\rangle \quad . \tag{7}$$

Although for $m \neq 1$ $a_m < c$, for some vector component the sum $\sum_{m=2}^{M} a_m|c_m\rangle$ may accumulate to such a large value as to affect the thresholded result $|b\rangle$. The problem would be avoided if the thresholding operation $s(\ )$ in the back layer of the BAM were to be replaced by another nonlinear operation which selects, from the linear combination

$$c|c_1\rangle + \sum_{m=2}^{M} a_m|c_m\rangle \tag{8}$$

the dominant label $|c_1\rangle$. The hypothetical device which performs this operation is here called the "Dominant Label Selector" (DLS)[11], and we call the resulting memory architecture "Selective Reflexive Memory" (SRM). With the back state selected as the dominant label $|c_1\rangle$, the back stroke gives $\langle f'| = s(\langle c_1|K) = s(P\langle 1|) = \langle 1|$, by the orthogonality of the labels $|c_m\rangle$. It follows[11] that the SRM gives perfect associative recall of the nearest stored data vector, for any number of vectors stored. Of course, the linear independence of the P-dimensional label vectors $|c_m\rangle$, m=1 to M, requires $P \geq M$.

The DLS must select, from a linear combination of orthonormal labels, the dominant label. A trivial case is obtained by choosing the

labels $|c_m\rangle$ as basis vectors $|u_m\rangle$, which have all components zero except for the mth component, which is unity. With this choice of labels, the

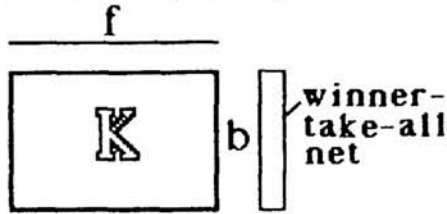

Fig.7. Simplest reflexive memory with DLS

DLS may be taken as a winner-take-all net W, as shown in Fig. 7. This case appears to be included in Adaptive Resonance Theory (ART)[12] as a special simplified case. A relationship between the ordinary BAM and ART was pointed out by Kosko[5]. As in ART, there is considerable fault sensitivity in this memory, because the stored data vectors appear in the connection matrix as rows.

A memory with better fault tolerance may be obtained by using orthogonal labels other than basis vectors. The DLS can then be taken as an orthogonal transformation G followed by a winner-take-all net, as shown in Fig. 8. G is to be chosen such that it transforms the labels $|c_m\rangle$

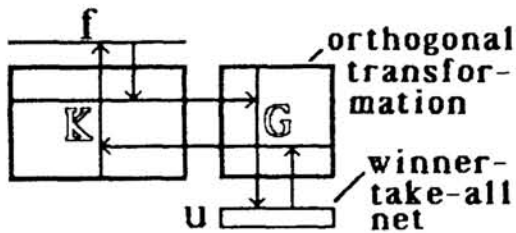

Fig. 8. Selective reflexive memory

into vectors proportional to the basis vectors $|u_m\rangle$. This can always be done by taking

$$G=\sum_{p=1}^{P}|u_p\rangle\langle c_p| \; , \qquad (9)$$

where the $|c_p\rangle$, p=1 to P, form a complete orthonormal set which contains the labels $|c_m\rangle$, m=1 to M. The neurons in the DLS serve as grandmother cells. Once a single winning cell has been activated, i.e., the state of the layer is a single basis vector, say $|u_1\rangle$, this vector must be passed back, after application of the transformation $G^{-1}$, such as to produce the label $|c_1\rangle$ at the back of the BAM. Since G is orthogonal, we have $G^{-1}=G^T$, so that the required inverse transformation may be accomplished simply by sending the basis vector back through the transformer; this gives

$$\langle u_1|G=\sum_{p=1}^{P}\langle u_1|u_p\rangle\langle c_p|=\langle c_1| \qquad . \qquad (10)$$

as required.

## HALF SRM

The SRM may be modified by deleting the thresholding operation in the front layer. The front neurons then have a linear output, which is reflected back through the SRM, as shown in Fig. 9. In this case, the

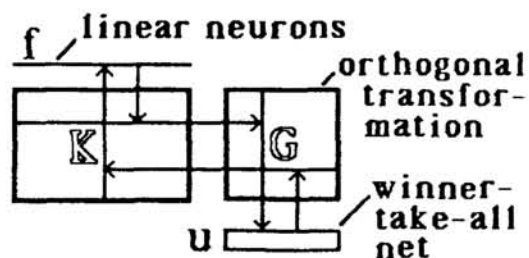

Fig. 9. Half SRM with linear
neurons in front layer

stored data vectors and the input data vectors may be taken as analog vectors, but we require all the stored vectors to have the same norm. The action of the SRM proceeds in the same way as described above, except that we now require the orthonormal labels to have unit norm. It follows that, just like the full SRM, the half SRM gives perfect associative recall to the nearest stored vector, for any number of stored vectors up to the dimension P of the labels. The latter condition is due to the fact that a P-dimensional vector space can at most contain P orthonormal vectors.

In the SRM the output transform G is introduced in order to improve the fault tolerance of the connection matrix K. This is accomplished at the cost of some fault sensitivity of G, the extent of which needs to be investigated. In this regard it is noted that in certain optical implementations of reflexive memories, such as Dana Anderson's resonator[1] and similar configurations[2,3], the transformation G is a Fourier transform, which is implemented simply as a lens. Such an implementation is quite insentive to the common semiconductor damage mechanisms.

## EQUIVALENT AUTOASSOCIATIVE MEMORIES

Concatenation of the front and back state vectors allows description of the SRMs in terms of autoassociative memories. For the SRM which uses basis vectors as labels the corresponding autoassociative memory is shown in Fig. 10. This connection matrix structure was also proposed by Guest et. al.[13]. The winner-take-all net W needs to be

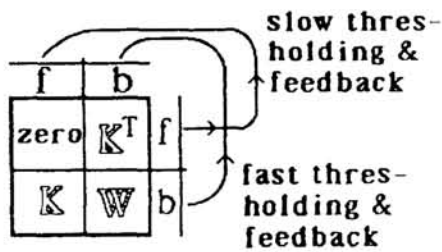

Fig. 10. Equivalent auto-
associative memory

given time to settle on a basis
vector state before the state |b⟩
can influence the front state |f⟩.
This may perhaps be achieved by
arranging the W network to have a
thresholding and feedback which
are fast compared with that of the
K network. An alternate method
may be to equip the W network
with an output gate which is
opened only after the W net has
settled. These arrangements
present a complication and cause a delay, which in some applications
may be inappropriate, and in others may be acceptable in a trade
between speed and memory density.

For the SRM with output transformer and orthonormal labels other
than basis vectors, a correspon-

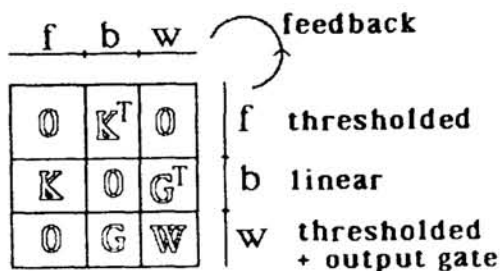

Fig. 11. Autoassociative memory
equivalent to SRM with transform

ding autoassociative memory may
be composed as shown in Fig.11.
An output gate in the w layer is
chosen as the device which
prevents the backstroke through
the BAM to take place before the
winner-take-al net has settled.
The same effect may perhaps be
achieved by choosing different
response times for the neuron
layers f and w. These matters
require investigation. Unless
the output transform G is already
required for other reasons, as in
some optical resonators, the DLS
with output transform is clumsy.
It would far better to combine
the transformer G and the net W
into a single network. To find
such a DLS should be considered
a challenge.

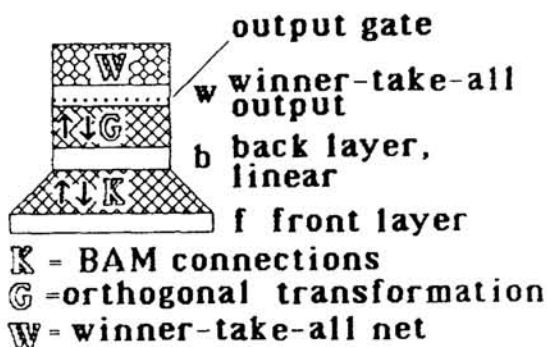

K = BAM connections
G =orthogonal transformation
W = winner-take-all net

Fig. 12. Structure of SRM

The work was partly supported by the Defense Advanced Research Projects Agency, ARPA order #5916, through Contract DAAH01-86-C -0968 with the U.S. Army Missile Command.

## REFERENCES

1. D. Z. Anderson, "Coherent optical eigenstate memory", Opt. Lett. 11, 56 (1986).

2. B. H. Soffer, G. J. Dunning, Y. Owechko, and E. Marom, "Associative holographic memory with feedback using phase-conjugate mirrors", Opt. Lett. 11, 118 (1986).

3. A. Yarriv and S. K. Wong, "Associative memories based on message-bearing optical modes in phase-conjugate resonators", Opt. Lett. 11, 186 (1986).

4. B. Kosko, "Adaptive Cognitive Processing", NSF Workshop for Neural Networks and Neuromorphic Systems, Boston, Mass., Oct. &-8, 1986.

5. B. Kosko, "Bidirectional Associative Memories", IEEE Trans. SMC, in press, 1987.

6. B. Kosko, "Adaptive Bidirectional Associative Memories", Appl. Opt., in press, 1987.

7. J. J. Hopfield, "Neural networks and physical systems with emergent collective computational abilities", Proc. Natl. Acad. Sci. USA 79, 2554 (1982).

8. P. A. M. Dirac, THE PRINCIPLES OF QUANTUM MECHANICS, Oxford, 1958.

9. T. Kohonen, "Correlation Matrix Memories", Helsinski University of Technology Report TKK-F-A130, 1970.

10. M. Harwit and N. J. A. Sloane, HADAMARD TRANSFORM OPTICS, Academic Press, New York, 1979.

11. H. G. Loos, "Adaptive Stochastic Content-Addressable Memory", Final Report, ARPA Order 5916, Contract DAAH01-86-C-0968, March 1987.

12. G. A. Carpenter and S. Grossberg, "A Massively Parallel Architecture for a Self-Organizing Neural Pattern Recognition Machine", Computer Vision, Graphics, and Image Processing, 37, 54 (1987).

13. R. D. TeKolste and C. C. Guest , "Optical Cohen-Grossberg System with All-Optical Feedback", IEEE First Annual International Conference on Neural Networks, San Diego, June 21-24, 1987.